# Exact Convex Confidence-Weighted Learning

**Koby Crammer    Mark Dredze    Fernando Pereira**[*]
Department of Computer and Information Science , University of Pennsylvania
Philadelphia, PA 19104
{crammer,mdredze,pereira}@cis.upenn.edu

## Abstract

Confidence-weighted (CW) learning [6], an online learning method for linear clas-
sifiers, maintains a Gaussian distributions over weight vectors, with a covariance
matrix that represents uncertainty about weights and correlations. Confidence
constraints ensure that a weight vector drawn from the hypothesis distribution
correctly classifies examples with a specified probability. Within this framework,
we derive a new convex form of the constraint and analyze it in the mistake bound
model. Empirical evaluation with both synthetic and text data shows our version of
CW learning achieves lower cumulative and out-of-sample errors than commonly
used first-order and second-order online methods.

## 1   Introduction

Online learning methods for linear classifiers, such as the perceptron and passive-aggressive (PA)
algorithms [4], have been thoroughly analyzed and are widely used. However, these methods do not
model the strength of evidence for different weights arising from differences in the use of features
in the data, which can be a serious issue in text classification, where weights of rare features should
be trusted less than weights of frequent features.

Confidence-weighted (CW) learning [6], motivated by PA learning, explicitly models classifier
weight uncertainty with a full multivariate Gaussian distribution over weight vectors. The PA ge-
ometrical margin constraint is replaced by the probabilistic constraint that a classifier drawn from
the distribution should, with high probability, classify correctly the next example. While Dredze
*et al.* [6] explained CW learning in terms of the standard deviation of the margin induced by the
hypothesis Gaussian, in practice they used the margin variance to make the problem convex. In this
work, we use their original constraint but maintain convexity, yielding experimental improvements.
Our primary contributions are a mistake-bound analysis [11] and comparison with related methods.

We emphasize that this work focuses on the question of uncertainty about feature weights, not on
confidence in predictions. In large-margin classification, the margin's magnitude for an instance
is sometimes taken as a proxy for prediction confidence for that instance, but that quantity is not
calibrated nor is it connected precisely to a measure of weight uncertainty. Bayesian approaches to
linear classification, such as Bayesian logistic regression [9], use a simple mathematical relationship
between weight uncertainty and prediction uncertainty, which unfortunately cannot be computed
exactly. CW learning preserves the convenient computational properties of PA algorithms while
providing a precise connection between weight uncertainty and prediction confidence that has led to
weight updates that are more effective in practice [6, 5].

We begin with a review of the CW approach, then show that the constraint can be expressed in a
convex form, and solve it to obtain a new CW algorithm. We also examine a dual representation
that supports kernelization. Our analysis provides a mistake bound and indicates that the algorithm
is invariant to initialization. Simulations show that our algorithm improves over first-order methods

---

[*]Current affiliation: Google, Mountain View, CA 94043, USA.

(perceptron and PA) as well as other second order methods (second-order perceptron). We conclude with a review of related work.

## 2  Confidence-Weighted Linear Classification

The CW binary-classifier learner works in rounds. On round $i$, the algorithm applies its current linear classification rule $h_{\boldsymbol{w}}(\boldsymbol{x}) = \text{sign}(\boldsymbol{w} \cdot \boldsymbol{x})$ to an instance $\boldsymbol{x}_i \in \mathbb{R}^d$ to produce a prediction $\hat{y}_i \in \{-1, +1\}$, receives a true label $y_i \in \{-1, +1\}$ and suffers a loss $\ell(y_i, \hat{y}_i)$. The rule $h_{\boldsymbol{w}}$ can be identified with $\boldsymbol{w}$ up to a scaling, and we will do so in what follows since our algorithm will turn out to be scale-invariant. As usual, we define the *margin* of an example on round $i$ as $m_i = y_i(\boldsymbol{w}_i \cdot \boldsymbol{x}_i)$, where positive sign corresponds to a correct prediction.

CW classification captures the notion of confidence in the weights of a linear classifier with a probability density on classifier weight vectors, specifically a Gaussian distribution with mean $\boldsymbol{\mu} \in \mathbb{R}^d$ and covariance matrix $\Sigma \in \mathbb{R}^{d \times d}$. The values $\mu_p$ and $\Sigma_{p,p}$ represent knowledge of and confidence in the weight for feature $p$. The smaller $\Sigma_{p,p}$, the more confidence we have in the mean weight value $\mu_p$. Each covariance term $\Sigma_{p,q}$ captures our knowledge of the interaction between features $p$ and $q$.

In the CW model, the traditional signed margin is the mean of the induced univariate Gaussian random variable

$$M \sim \mathcal{N}\left(y(\boldsymbol{\mu} \cdot \boldsymbol{x}), \boldsymbol{x}^\top \Sigma \boldsymbol{x}\right) . \tag{1}$$

This probabilistic model can be used for prediction in different ways. Here, we use the average weight vector $\text{E}[\boldsymbol{w}] = \boldsymbol{\mu}$, analogous to Bayes point machines [8]. The information captured by the covariance $\Sigma$ is then used just to adjust training updates.

## 3  Update Rule

The CW update rule of Dredze *et al.* [6] makes the smallest adjustment to the distribution that ensures the probability of correct prediction on instance $i$ is no smaller than the confidence hyperparameter $\eta \in [0, 1]$: $\Pr[y_i(\boldsymbol{w} \cdot \boldsymbol{x}_i) \geq 0] \geq \eta$. The magnitude of the update is measured by its KL divergence to the previous distribution, yielding the following constrained optimization:

$$(\boldsymbol{\mu}_{i+1}, \Sigma_{i+1}) = \arg\min_{\boldsymbol{\mu},\Sigma} D_{\text{KL}}\left(\mathcal{N}(\boldsymbol{\mu}, \Sigma) \,\|\, \mathcal{N}(\boldsymbol{\mu}_i, \Sigma_i)\right) \quad \text{s.t.} \quad \Pr[y_i(\boldsymbol{w} \cdot \boldsymbol{x}_i) \geq 0] \geq \eta . \tag{2}$$

They rewrite the above optimization in terms of the standard deviation as:

$$\min \frac{1}{2}\left\{ \log\left(\frac{\det\Sigma_i}{\det\Sigma}\right) + \text{Tr}\left(\Sigma_i^{-1}\Sigma\right) + (\boldsymbol{\mu}_i - \boldsymbol{\mu})^\top \Sigma_i^{-1}(\boldsymbol{\mu}_i - \boldsymbol{\mu}) \right\} \text{ s.t. } y_i(\boldsymbol{\mu} \cdot \boldsymbol{x}_i) \geq \phi\sqrt{\boldsymbol{x}_i^\top \Sigma \boldsymbol{x}_i} . \tag{3}$$

Unfortunately, while the constraint of this problem is linear in $\boldsymbol{\mu}$, it is not convex in $\Sigma$. Dredze *et al.* [6, eq. (7)] circumvented that lack of convexity by removing the square root from the right-hand-size of the constraint, which yields the variance. However, we found that the original optimization can be preserved while maintaining convexity with a change of variable. Since $\Sigma$ is positive semidefinite (PSD), it can be written as $\Sigma = \Upsilon^2$ with $\Upsilon = Q\text{diag}(\lambda_1^{1/2}, \ldots, \lambda_d^{1/2})Q^\top$ where $Q$ is orthonormal and $\lambda_1, \ldots, \lambda_d$ are the eigenvalues of $\Sigma$; $\Upsilon$ is thus also PSD. This change yields the following convex optimization with a convex constraint in $\boldsymbol{\mu}$ and $\Upsilon$ simultaneously:

$$(\boldsymbol{\mu}_{i+1}, \Upsilon_{i+1}) = \arg\min \frac{1}{2}\log\left(\frac{\det\Upsilon_i^2}{\det\Upsilon^2}\right) + \frac{1}{2}\text{Tr}\left(\Upsilon_i^{-2}\Upsilon^2\right) + \frac{1}{2}(\boldsymbol{\mu}_i - \boldsymbol{\mu})^\top \Upsilon_i^{-2}(\boldsymbol{\mu}_i - \boldsymbol{\mu})$$

$$\text{s.t.} \quad y_i(\boldsymbol{\mu} \cdot \boldsymbol{x}_i) \geq \phi\|\Upsilon \boldsymbol{x}_i\| \quad , \quad \Upsilon \text{ is PSD} . \tag{4}$$

We call our algorithm CW-Stdev and the original algorithm of Dredze *et al.* CW-Var.

### 3.1  Closed-Form Update

While standard optimization techniques can solve the convex program (4), we favor a closed-form solution. Omitting the PSD constraint for now, we obtain the Lagrangian for (4),

$$\mathcal{L} = \frac{1}{2}\left[\log\left(\frac{\det\Upsilon_i^2}{\det\Upsilon^2}\right) + \text{Tr}\left(\Upsilon_i^{-2}\Upsilon^2\right) + (\boldsymbol{\mu}_i - \boldsymbol{\mu})^\top \Upsilon_i^{-2}(\boldsymbol{\mu}_i - \boldsymbol{\mu})\right] + \alpha\left(-y_i(\boldsymbol{\mu} \cdot \boldsymbol{x}_i) + \phi\|\Upsilon \boldsymbol{x}_i\|\right) \tag{5}$$

**Input parameters**   $a > 0 \, ; \eta \in [0.5, 1]$

**Initialize**   $\boldsymbol{\mu}_1 = \mathbf{0}$ , $\Sigma_1 = aI$ , $\phi = \Phi^{-1}(\eta)$ , $\psi = 1 + \phi^2/2$ , $\xi = 1 + \phi^2$ .

**For** $i = 1, \ldots, n$

- Receive a training example $\boldsymbol{x}_i \in \mathbb{R}^d$
- Compute Gaussian margin distribution $M_i \sim \mathcal{N}\big( (\boldsymbol{\mu}_i \cdot \boldsymbol{x}_i) , (\boldsymbol{x}_i^\top \Sigma_i \boldsymbol{x}_i) \big)$
- Receive true label $y_i$ and compute

$$v_i = \boldsymbol{x}_i^\top \Sigma_i \boldsymbol{x}_i \; , \;\; m_i = y_i (\boldsymbol{\mu}_i \cdot \boldsymbol{x}_i) \,(11) \quad , \;\; u_i = \frac{1}{4}\left( -\alpha v_i \phi + \sqrt{\alpha^2 v_i^2 \phi^2 + 4 v_i} \right)^2 \quad (12)$$

$$\alpha_i = \max\left\{ 0, \frac{1}{v_i \xi}\left( -m_i \psi + \sqrt{m_i^2 \frac{\phi^4}{4} + v_i \phi^2 \xi} \right) \right\} (14) \quad , \;\; \beta_i = \frac{\alpha_i \phi}{\sqrt{u_i} + v_i \alpha_i \phi} \qquad (22)$$

- Update   $\boldsymbol{\mu}_{i+1} = \boldsymbol{\mu}_i + \alpha_i y_i \Sigma_i \boldsymbol{x}_i$

$$\Sigma_{i+1} = \Sigma_i - \beta_i \Sigma_i \boldsymbol{x}_i \boldsymbol{x}_i^\top \Sigma_i \qquad\qquad\qquad \text{(full)} \qquad\qquad (10)$$

$$\Sigma_{i+1} = \left( \Sigma_i^{-1} + \alpha_i \phi u_i^{-\frac{1}{2}} \text{diag}^2(\boldsymbol{x}_i) \right)^{-1} \qquad\qquad \text{(diag)} \qquad\qquad (15)$$

**Output**   Gaussian distribution $\mathcal{N}\big( \boldsymbol{\mu}_{n+1}, \Sigma_{n+1} \big)$.

---

Figure 1: The CW-Stdev algorithm. The numbers in parentheses refer to equations in the text.

At the optimum, it must be that

$$\frac{\partial}{\partial \boldsymbol{\mu}} \mathcal{L} = \Upsilon_i^{-2}(\boldsymbol{\mu} - \boldsymbol{\mu}_i) - \alpha y_i \boldsymbol{x}_i = 0 \qquad \Rightarrow \qquad \boldsymbol{\mu}_{i+1} = \boldsymbol{\mu}_i + \alpha y_i \Upsilon_i^2 \boldsymbol{x}_i \, , \qquad (6)$$

where we assumed that $\Upsilon_i$ is non-singular (PSD). At the optimum, we must also have,

$$\frac{\partial}{\partial \Upsilon} \mathcal{L} = -\Upsilon^{-1} + \frac{1}{2}\Upsilon_i^{-2}\Upsilon + \frac{1}{2}\Upsilon\Upsilon_i^{-2} + \alpha\phi \frac{\boldsymbol{x}_i \boldsymbol{x}_i^\top \Upsilon}{2\sqrt{\boldsymbol{x}_i^\top \Upsilon^2 \boldsymbol{x}_i}} + \alpha\phi \frac{\Upsilon \boldsymbol{x}_i \boldsymbol{x}_i^\top}{2\sqrt{\boldsymbol{x}_i^\top \Upsilon^2 \boldsymbol{x}_i}} = 0 \, , \qquad (7)$$

from which we obtain the implicit-form update

$$\Upsilon_{i+1}^{-2} = \Upsilon_i^{-2} + \alpha\phi \frac{\boldsymbol{x}_i \boldsymbol{x}_i^\top}{\sqrt{\boldsymbol{x}_i^\top \Upsilon_{i+1}^2 \boldsymbol{x}_i}} \, . \qquad (8)$$

Conveniently, these updates can be expressed in terms of the covariance matrix [1] :

$$\boldsymbol{\mu}_{i+1} = \boldsymbol{\mu}_i + \alpha y_i \Sigma_i \boldsymbol{x}_i \qquad , \qquad \Sigma_{i+1}^{-1} = \Sigma_i^{-1} + \alpha\phi \frac{\boldsymbol{x}_i \boldsymbol{x}_i^\top}{\sqrt{\boldsymbol{x}_i^\top \Sigma_{i+1} \boldsymbol{x}_i}} \, . \qquad (9)$$

We observe that (9) computes $\Sigma_{i+1}^{-1}$ as the sum of a rank-one PSD matrix and $\Sigma_i^{-1}$ . Thus, if $\Sigma_i^{-1}$ has strictly positive eigenvalues, so do $\Sigma_{i+1}^{-1}$ and $\Sigma_{i+1}$ . Thus, $\Sigma_i$ and $\Upsilon_i$ are indeed PSD non-singular, as assumed above.

### 3.2  Solving for the Lagrange Multiplier $\alpha$

We now determine the value of the Lagrange multiplier $\alpha$ and make the covariance update explicit. We start by computing the inverse of (9) using the Woodbury identity [14, Eq. 135] to get

$$\Sigma_{i+1} = \left( \Sigma_i^{-1} + \alpha\phi \frac{\boldsymbol{x}_i \boldsymbol{x}_i^\top}{\sqrt{\boldsymbol{x}_i^\top \Sigma_{i+1} \boldsymbol{x}_i}} \right)^{-1} = \Sigma_i - \Sigma_i \boldsymbol{x}_i \left( \frac{\alpha\phi}{\sqrt{\boldsymbol{x}_i^\top \Sigma_{i+1}\boldsymbol{x}_i} + \boldsymbol{x}_i^\top \Sigma_i \boldsymbol{x}_i \alpha\phi} \right) \boldsymbol{x}_i^\top \Sigma_i \, . \;\; (10)$$

Let

$$u_i = \boldsymbol{x}_i^\top \Sigma_{i+1} \boldsymbol{x}_i \quad , \quad v_i = \boldsymbol{x}_i^\top \Sigma_i \boldsymbol{x}_i \quad , \quad m_i = y_i (\boldsymbol{\mu}_i \cdot \boldsymbol{x}_i) \, . \qquad (11)$$

Multiplying (10) by $\boldsymbol{x}_i^\top$ (left) and $\boldsymbol{x}_i$ (right) we get $u_i = v_i - v_i \left( \frac{\alpha\phi}{\sqrt{u_i}+v_i\alpha\phi} \right) v_i$ , which can be solved for $u_i$ to obtain

$$\sqrt{u_i} = \frac{-\alpha v_i \phi + \sqrt{\alpha^2 v_i^2 \phi^2 + 4v_i}}{2} \ . \tag{12}$$

The KKT conditions for the optimization imply that either $\alpha = 0$ and no update is needed, or the constraint (4) is an equality after the update. Using the equality version of (4) and Eqs. (9,10,11,12) we obtain $m_i + \alpha v_i = \phi \frac{-\alpha v_i \phi + \sqrt{\alpha^2 v_i^2 \phi^2 + 4v_i}}{2}$ , which can be rearranged into a quadratic equation in $\alpha$: $\alpha^2 v_i^2 \left(1 + \phi^2\right) + 2\alpha m_i v_i \left(1 + \frac{\phi^2}{2}\right) + \left(m_i^2 - v_i \phi^2\right) = 0$ . The smaller root of this equation is always negative and thus not a valid Lagrange multiplier. We use the following abbreviations for writing the larger root $\gamma_i$: $\psi = 1 + \phi^2/2$ ; $\xi = 1 + \phi^2$ . The larger root is then

$$\gamma_i = \frac{-m_i v_i \psi + \sqrt{m_i^2 v_i^2 \psi^2 - v_i^2 \psi \left(m_i^2 - v_i \phi^2\right)}}{v_i^2 \psi} \ . \tag{13}$$

The constraint (4) is satisfied before the update if $m_i - \phi\sqrt{v_i} \geq 0$. If $m_i \leq 0$, then $m_i \leq \phi\sqrt{v_i}$ and from (13) we have that $\gamma_i > 0$. If, instead, $m_i \geq 0$, then, again by (13), we have

$$\gamma_i > 0 \Leftrightarrow m_i v_i \psi < \sqrt{m_i^2 v_i^2 \psi^2 - v_i^2 \psi \left(m_i^2 - v_i \phi^2\right)} \Leftrightarrow m_i < \phi v_i \ .$$

From the KKT conditions, either $\alpha_i = 0$ or (3) is satisfied as an equality and $\alpha_i = \gamma_i > 0$. We summarize the discussion in the following lemma:

**Lemma 1** *The solution of* (13) *satisfies the KKT conditions, that is either $\alpha_i \geq 0$ or the constraint of* (3) *is satisfied before the update with the parameters $\boldsymbol{\mu}_i$ and $\Sigma_i$.*

We obtain the final form of $\alpha_i$ by simplifying (13) together with Lemma 1,

$$\max\left\{0, \frac{1}{v_i} \frac{-m_i\psi + \sqrt{m_i^2 \frac{\phi^4}{4} + v_i \phi^2 \xi}}{\xi}\right\} \ . \tag{14}$$

To summarize, after receiving the correct label $y_i$ the algorithm checks whether the probability of a correct prediction under the current parameters is greater than a confidence threshold $\eta = \Phi(\phi)$. If so, it does nothing. Otherwise it performs an update as described above. We initialize $\boldsymbol{\mu}_1 = \mathbf{0}$ and $\Sigma_1 = aI$ for some $a > 0$. The algorithm is summarized in Fig. 1.

Two comments are in order. First, if $\eta = 0.5$, then from Eq. (9) we see that only $\boldsymbol{\mu}$ will be updated, not $\Sigma$, because $\phi = 0 \Leftrightarrow \eta = 0.5$. In this case the covariance $\Sigma$ parameter does not influence the decision, only the mean $\boldsymbol{\mu}$. Furthermore, for length-one input vectors, at the first round we have $\Sigma_1 = aI$, so the first-round constraint is $y_i (\boldsymbol{w}_i \cdot \boldsymbol{x}_i) \geq a \|\boldsymbol{x}_i\|^2 = a$, which is equivalent to the original PA update.

Second, the update described above yields full covariance matrices. However, sometimes we may prefer diagonal covariance matrices, which can be achieved by projecting the matrix $\Sigma_{i+1}$ that results from the update onto the set of diagonal matrices. In practice it requires setting all the off-diagonal elements to zero, leaving only the diagonal elements. In fact, if $\Sigma_i$ is diagonal then we only need to project $\boldsymbol{x}_i \boldsymbol{x}_i^\top$ to a diagonal matrix. We thus replace (9) with the following update,

$$\Sigma_{i+1}^{-1} = \Sigma_i^{-1} + \phi \frac{\alpha_i}{\sqrt{u_i}} \text{diag}^2\left(\boldsymbol{x}_i\right) \ , \tag{15}$$

where $\text{diag}^2\left(\boldsymbol{x}_i\right)$ is a diagonal matrix made from the squares of the elements of $\boldsymbol{x}_i$ on the diagonal. Note that for diagonal matrices there is no need to use the Woodbury equation to compute the inverse, as it can be computed directly element-wise. We use CW-Stdev (or CW-Stdev-full) to refer to the full-covariance algorithm, and CW-Stdev-diag to refer to the diagonal-covariance algorithm.

Finally, the following property of our algorithm shows that it can be used with Mercer kernels:

**Theorem 2 (Representer Theorem)** *The mean $\boldsymbol{\mu}_i$ and covariance $\Sigma_i$ parameters computed by the algorithm in Fig. 1 can be written as linear combinations of the input vectors with coefficients that depend only on inner products of input vectors:*

$$\Sigma_i = \sum_{p,q=1}^{i-1} \pi_{p,q}^{(i)} \boldsymbol{x}_p \boldsymbol{x}_q^\top + aI \qquad , \qquad \boldsymbol{\mu}_i = \sum_p^{i-1} \nu_p^{(i)} \boldsymbol{x}_p \ . \tag{16}$$

The proof, given in the appendix, is a simple induction.

## 4 Analysis

We analyze CW-Stdev in two steps. First, we show that performance does not depend on initialization and then we compute a bound on the number of mistakes that the algorithm makes.

### 4.1 Invariance to Initialization

The algorithm in Fig. 1 uses a predefined parameter $a$ to initialize the covariance matrix. Since the decision to update depends on the covariance matrix, which implicitly depends on $a$ through $\alpha_i$ and $v_i$, one may assume that $a$ effects performance. In fact the number of mistakes is *independent* of $a$, i.e. the constraint of (3) is invariant to scaling. Specifically, if it holds for mean and covariance parameters $\boldsymbol{\mu}$ and $\Sigma$, it holds also for the scaled parameters $c\boldsymbol{\mu}$ and $c^2\Sigma$ for any $c > 0$. The following lemma states that the scaling is controlled by $a$. Thus, we can always initialize the algorithm with a value of $a = 1$. If, in addition to predictions, we also need the distribution over weight vectors, the scale parameter $a$ should be calibrated.

**Lemma 3** *Fix a sequence of examples $(\boldsymbol{x}_1, \boldsymbol{y}_1) \dots (\boldsymbol{x}_n, \boldsymbol{y}_n)$. Let $\Sigma_i, \boldsymbol{\mu}_i, m_i, v_i, \alpha_i, u_i$ be the quantities obtained throughout the execution of the algorithm described in Fig. 1 initialized with $(\mathbf{0}, I)$ $(a = 1)$. Let also $\tilde{\Sigma}_i, \tilde{\boldsymbol{\mu}}_i, \tilde{m}_i, \tilde{v}_i, \tilde{\alpha}_i, \tilde{u}_i$ be the corresponding quantities obtained throughout the execution of the algorithm, with an alternative initialization of $(\mathbf{0}, aI)$ (for some $a > 0$). The following relations between the two set of quantities hold:*

$$\tilde{m}_i = \sqrt{a} m_i \ , \ \ \tilde{v}_i = a v_i \ , \ \ \tilde{\alpha}_i = \frac{1}{\sqrt{a}} \alpha_i \ , \ \ \tilde{\boldsymbol{\mu}}_i = \sqrt{a} \boldsymbol{\mu}_i \ , \ \ \tilde{u}_i = a u_i \ , \ \ \tilde{\Sigma}_i = a \Sigma_i \ . \tag{17}$$

**Proof sketch:** The proof proceeds by induction. The initial values of these quantities clearly satisfy the required equalities. For the induction step we assume that (17) holds for some $i$ and show that these identities also hold for $i + 1$ using Eqs. (9,14,11,12) .                                    ∎
From the lemma we see that the quantity $\tilde{m}_i / \sqrt{\tilde{v}_i} = m_i / \sqrt{v_i}$ is invariant to $a$. Therefore, the behavior of the algorithm in general, and its updates and mistakes in particular, are independent to the choice of $a$. Therefore, we assume $a = 1$ in what follows.

### 4.2 Analysis in the Mistake Bound Model

The main theorem of the paper bounds the number of mistakes made by CW-Stdev.

**Theorem 4** *Let $(\boldsymbol{x}_1, \boldsymbol{y}_1) \dots (\boldsymbol{x}_n, \boldsymbol{y}_n)$ be an input sequence for the algorithm of Fig. 1, initialized with $(\mathbf{0}, I)$, with $\boldsymbol{x}_i \in \mathbb{R}^d$ and $\boldsymbol{y}_i \in \{-1, +1\}$ . Assume there exist $\boldsymbol{\mu}^*$ and $\Sigma^*$ such that for all $i$ for which the algorithm made an update ($\alpha_i > 0$),*

$$\boldsymbol{\mu}^{*\top} \boldsymbol{x}_i y_i \geq \boldsymbol{\mu}_{i+1}^\top \boldsymbol{x}_i y_i \quad and \quad \boldsymbol{x}_i^\top \Sigma^* \boldsymbol{x}_i \leq \boldsymbol{x}_i^\top \Sigma_{i+1} \boldsymbol{x}_i \quad . \tag{18}$$

*Then the following holds:*

$$no. \ mistakes \leq \sum_i \alpha_i^2 v_i \leq \frac{1 + \phi^2}{\phi^2} \left( -\log \det \Sigma^* + \mathrm{Tr}\left(\Sigma^*\right) + \boldsymbol{\mu}^{*\top} \Sigma_{n+1}^{-1} \boldsymbol{\mu}^* - d \right) \tag{19}$$

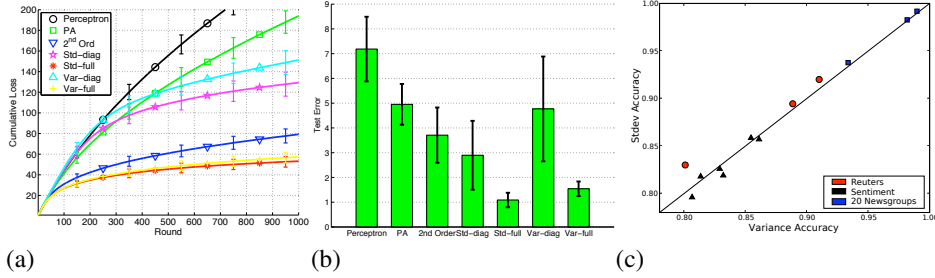

(a)                                    (b)                                    (c)

Figure 2: (a) The average and standard deviation of the cumulative number of mistakes for seven algorithms. (b) The average and standard deviation of test error (%) over unseen data for the seven algorithms. (c) Comparison between CW-Stdev-diag and CW-Var-diag on text classification.

The proof is given in the appendix.

The above bound depends on an output of the algorithm, $\Sigma_{n+1}$, similar to the bound for the second-order perceptron [3]. The two conditions (18) imply linear separability of the input sequence by $\boldsymbol{\mu}^*$:

$$\boldsymbol{\mu}^{*\top}\boldsymbol{x}_i y_i \overset{(18)}{\geq} \boldsymbol{\mu}_{i+1}^{\top}\boldsymbol{x}_i y_i \overset{(4)}{\geq} \phi\sqrt{\boldsymbol{x}_i^{\top}\Sigma_{i+1}\boldsymbol{x}_i} \overset{(18)}{\geq} \boldsymbol{x}_i^{\top}\Sigma^*\boldsymbol{x}_i \geq \min_i \boldsymbol{x}_i^{\top}\Sigma^*\boldsymbol{x}_i > 0 \, ,$$

where the superscripts in parentheses refer to the inequalities used. From (10), we observe that $\Sigma_{i+1} \preceq \Sigma_i$ for all $i$, so $\Sigma_{n+1} \preceq \Sigma_{i+1} \preceq \Sigma_1 = I$ for all $i$. Therefore, the conditions on $\Sigma^*$ in (18) are satisfied by $\Sigma^* = \Sigma_{n+1}$. Furthermore, if $\boldsymbol{\mu}^*$ satisfies the stronger conditions $y_i(\boldsymbol{\mu}^* \cdot \boldsymbol{x}_i) \geq \|\boldsymbol{x}_i\|$, from $\Sigma_{i+1} \preceq I$ above it follows that

$$(\phi\boldsymbol{\mu}^*)^{\top}\boldsymbol{x}_i y_i \geq \phi\|\boldsymbol{x}_i\| = \phi\sqrt{\boldsymbol{x}_i^{\top} I \boldsymbol{x}_i} \geq \phi\sqrt{\boldsymbol{x}_i^{\top}\Sigma_{i+1}\boldsymbol{x}_i} = \boldsymbol{\mu}_{i+1}^{\top}\boldsymbol{x}_i y_i \, ,$$

where the last equality holds since we assumed that an update was made for the $i$th example. In this situation, the bound becomes

$$\frac{\phi^2+1}{\phi^2}\left(-\log\det\Sigma_{n+1} + \mathrm{Tr}\left(\Sigma_{n+1}\right) - d\right) + (\phi^2+1)\left(\boldsymbol{\mu}^{*\top}\Sigma_{n+1}^{-1}\boldsymbol{\mu}^*\right) \, .$$

The quantity $\boldsymbol{\mu}^{*\top}\Sigma_{n+1}^{-1}\boldsymbol{\mu}^*$ in this bound is analogous to the quantity $R^2\|\boldsymbol{\mu}^*\|^2$ in the perceptron bound [13], except that the norm of the examples does not come in explicitly as the radius $R$ of the enclosing ball, but implicitly through the fact that $\Sigma_{n+1}^{-1}$ is a sum of example outer products (9). In addition, in this version of the bound we impose a margin of 1 under the condition that examples have unit norm, whereas in the perceptron bound, the margin of 1 is for examples with arbitrary norm. This follows from the fact that (4) is invariant to the norm of $\boldsymbol{x}_i$.

## 5   Empirical Evaluation

We illustrate the benefits of CW-Stdev with synthetic data experiments. We generated $1,000$ points in $\mathbb{R}^{20}$ where the first two coordinates were drawn from a $45°$ rotated Gaussian distribution with standard deviation 1. The remaining 18 coordinates were drawn from independent Gaussian distributions $\mathcal{N}(0,2)$. Each point's label depended on the first two coordinates using a separator parallel to the long axis of the ellipsoid, yielding a linearly separable set (Fig. 3(top)). We evaluated five online learning algorithms: the perceptron [16], the passive-aggressive (PA) algorithm [4], the second-order perceptron (SOP) [3], CW-Var-diag, CW-Var-full [6], CW-Stdev-diag and CW-Stdev-full. All algorithm parameters were tuned over $1,000$ runs.

Fig. 2(a) shows the average cumulative mistakes for each algorithm; error bars indicate one unit of standard deviation. Clearly, second-order algorithms, which all made fewer than $80$ mistakes, outperform the first-order ones, which made at least $129$ mistakes. Additionally, CW-Var makes more mistakes than CW-Stdev: 8% more in the diagonal case and 17% more in the full. The diagonal methods performed better than the first order methods, indicating that while they do not use any

second-order information, they capture additional information for single features. For each repetition, we evaluated the resulting classifiers on $10,000$ unseen test examples (Fig. 2(b)). Averaging improved the first-order methods. The second-order methods outperform the first-order methods, and CW-Stdev outperforms all the other methods. Also, the full case is less sensitive across runs.

The Gaussian distribution over weight vectors after $50$ rounds is represented in Fig. 3(bot). The 20 dimensions of the version space are grouped into 10 pairs, the first containing the two meaningful features. The dotted segment represents the first two coordinates of possible representations of the true hyperplane in the positive quadrant. Clearly, the corresponding vectors are orthogonal to the hyperplane shown in Fig. 3(top). The solid black ellipsoid represents the first two significant feature weights; it does not yet lie of the dotted segment because the algorithm has not converged. Nevertheless, the long axis is already parallel to the true set of possible weight vectors. The axis perpendicular to the weight-vector set is very small, showing that there is little freedom in that direction. The remaining nine ellipsoids represent the covariance of pairs of noise features. Those ellipsoids are close to circular and have centers close to the origin, indicating that the corresponding feature weights should be near zero but without much confidence.

**NLP Evaluation:** We compared CW-Stdev-diag with CW-Var-diag, which beat many state of the art algorithms on 12 NLP datasets [6]. We followed the same evaluation setting using 10-fold cross validation and the same splits for both algorithms. Fig. 2(c) compares the accuracy on test data of each algorithm; points above the line represent improvements of CW-Stdev over CW-Var. Stdev improved on eight of the twelve datasets and, while the improvements are not significant, they show the effectiveness of our algorithm on real world data.

## 6 Related Work

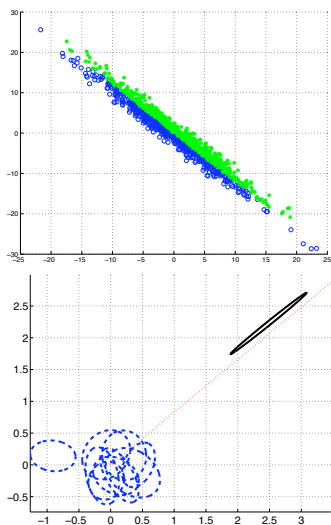

Figure 3: Top : Plot of the two informative features of the synthetic data. Bottom: Feature weight distributions of CW-Stdev-full after $50$ examples.

Online additive algorithms have a long history, from with the perceptron [16] to more recent methods [10, 4]. Our update has a more general form, in which the input vector $x_i$ is linearly transformed using the covariance matrix, both rotating the input and assigning weight specific learning rates. Weight-specific learning rates appear in neural-network learning [18], although they do not model confidence based on feature variance.

The second order perceptron (SOP) [3] demonstrated that second-order information can improve on first-order methods. Both SOP and CW maintain second-order information. SOP is mistake driven while CW is passive-aggressive. SOP uses the current instance in the correlation matrix for prediction while CW updates after prediction. A variant of CW-Stdev similar to SOP follows from our derivation if we fix the Lagrange multiplier in (5) to a predefined value $\alpha_i = \alpha$, omit the square root, and use a gradient-descent optimization step. Fundamentally, CW algorithms have a probabilistic motivation, while the SOP is geometric: replace the ball around an example with a refined ellipsoid. Shivaswamy and Jebara [17] used a similar motivation in batch learning.

Ensemble learning shares the idea of combining multiple classifiers. Gaussian process classification (GPC) maintains a Gaussian distribution over weight vectors (primal) or over regressor values (dual). Our algorithm uses a different update criterion than the standard GPC Bayesian updates [15, Ch.3], avoiding the challenge of approximating posteriors. Bayes point machines [8] maintain a collection of weight vectors consistent with the training data, and use the single linear classifier which best represents the collection. Conceptually, the collection is a non-parametric distribution over the weight vectors. Its online version [7] maintains a finite number of weight-vectors which are updated simultaneously. The rele-

vance vector machine [19] incorporates probabilistic models into the dual formulation of SVMs. As in our work, the dual parameters are random variables distributed according to a diagonal Gaussian with example specific variance. The weighted-majority [12] algorithm and later improvements [2] combine the output of multiple arbitrary classifiers, maintaining a multinomial distribution over the experts. We assume linear classifiers as experts and maintain a Gaussian distribution over their weight vectors.

# 7 Conclusion

We presented a new confidence-weighted learning method for linear classifier based on the standard deviation. We have shown that the algorithm is invariant to scaling and we provided a mistake-bound analysis. Based on both synthetic and NLP experiments, we have shown that our method improves upon recent first and second order methods. Our method also improves on previous CW algorithms. We are now investigating special cases of CW-Stdev for problems with very large numbers of features, multi-class classification, and batch training.

## Footnotes

[1]Furthermore, writing the Lagrangian of (3) and solving it would yield the same solution as Eqns. (9). Thus the optimal solution of both (3) and (4) are the same.

# References

[1] Y. Censor and S.A. Zenios. *Parallel Optimization: Theory, Algorithms, and Applications*. Oxford University Press, New York, NY, USA, 1997.

[2] N. Cesa-Bianchi, Y. Freund, D. Haussler, D. P. Helmbold, R. E. Schapire, and M. K. Warmuth. How to use expert advice. *Journal of the Association for Computing Machinery*, 44(3):427–485, May 1997.

[3] Nicoló Cesa-Bianchi, Alex Conconi, and Claudio Gentile. A second-order perceptron algorithm. *Siam Journal of Commutation*, 34(3):640–668, 2005.

[4] K. Crammer, O. Dekel, J. Keshet, S. Shalev-Shwartz, and Y. Singer. Online passive-aggressive algorithms. *Journal of Machine Learning Research*, 7:551–585, 2006.

[5] Mark Dredze and Koby Crammer. Active learning with confidence. In *ACL*, 2008.

[6] Mark Dredze, Koby Crammer, and Fernando Pereira. Confidence-weighted linear classification. In *International Conference on Machine Learning*, 2008.

[7] E. Harrington, R. Herbrich, J. Kivinen, J. Platt, and R.C. Williamson. Online bayes point machines. In *7th Pacific-Asia Conference on Knowledge Discovery and Data Mining (PAKDD)*, 2003.

[8] R. Herbrich, T. Graepel, and C. Campbell. Bayes point machines. *JMLR*, 1:245–279, 2001.

[9] T. Jaakkola and M. Jordan. A variational approach to bayesian logistic regression models and their extensions. In *Workshop on Artificial Intelligence and Statistics*, 1997.

[10] J. Kivinen and M. K. Warmuth. Exponentiated gradient versus gradient descent for linear predictors. *Information and Computation*, 132(1):1–64, January 1997.

[11] N. Littlestone. Learning when irrelevant attributes abound: A new linear-threshold algorithm. *Machine Learning*, 2:285–318, 1988.

[12] N. Littlestone and M. K. Warmuth. The weighted majority algorithm. *Information and Computation*, 108:212–261, 1994.

[13] A. B. J. Novikoff. On convergence proofs on perceptrons. In *Proceedings of the Symposium on the Mathematical Theory of Automata*, volume XII, pages 615–622, 1962.

[14] K. B. Petersen and M. S. Pedersen. The matrix cookbook, 2007.

[15] C. E. Rasmussen and C. K. I. Williams. *Gaussian Processes for Machine Learning*. The MIT Press, 2006.

[16] F. Rosenblatt. The perceptron: A probabilistic model for information storage and organization in the brain. *Psychological Review*, 65:386–407, 1958. (Reprinted in *Neurocomputing* (MIT Press, 1988).).

[17] P. Shivaswamy and T. Jebara. Ellipsoidal kernel machines. In *AISTATS*, 2007.

[18] Richard S. Sutton. Adapting bias by gradient descent: an incremental version of delta-bar-delta. In *Proceedings of the Tenth National Conference on Artificial Intelligence*, pages 171–176. MIT Press, 1992.

[19] M. E. Tipping. Sparse bayesian learning and the relevance vector machine. *Journal of Machine Learning Research*, 1:211–244, 2001.

[20] L. Xu, K. Crammer, and D. Schuurmans. Robust support vector machine training via convex outlier ablation. In *AAAI-2006*, 2006.

